# Fast Large-scale Mixture Modeling with Component-specific Data Partitions

**Bo Thiesson**[*]
Microsoft Research

**Chong Wang**[*†]
Princeton University

## Abstract

Remarkably easy implementation and guaranteed convergence has made the EM algorithm one of the most used algorithms for mixture modeling. On the downside, the E-step is linear in both the sample size and the number of mixture components, making it impractical for large-scale data. Based on the variational EM framework, we propose a fast alternative that uses component-specific data partitions to obtain a sub-linear E-step in sample size, while the algorithm still maintains provable convergence. Our approach builds on previous work, but is significantly faster and scales much better in the number of mixture components. We demonstrate this speedup by experiments on large-scale synthetic and real data.

## 1 Introduction

Probabilistic mixture modeling [7] has been widely used for density estimation and clustering applications. The Expectation-Maximization (EM) algorithm [4, 11] is one of the most used methods for this task for clear reasons – elegant formulation of an iterative procedure, ease of implementation, and guaranteed monotone convergence for the objective. On the other hand, the EM algorithm also has some acknowledged shortcomings. In particular, the E-step is linear in both the number of data points and the number of mixture components, and therefore computationally impractical for large-scale applications. Our work was motivated by a large-scale geo-spatial problem, demanding a mixture model of a customer base (a huge number of data points) for competing businesses (a large number mixture components), as the basis for site evaluation (where to locate a new store).

Several approximation schemes for EM have been proposed to address the scalability problem, e.g. [2, 12, 14, 10, 17, 16] , to mention a few. Besides [17, 16], none of these variants has both an E-step that is truly sub-linear in sample size and also enjoys provable convergence for a well-defined objective function. More details are discussed in Section 5. Our work is inspired by the "chunky EM" algorithm in [17, 16], a smart application of the variational EM framework [11], where a lower bound on the objective function increases at each iteration and convergence is guaranteed.

An E-step in standard EM calculates expected sufficient statistics under mixture-component membership probabilities calculated for each individual data point given the most recent model estimate. The variational EM framework alters the E-step to use sufficient statistics calculated under a variational distribution instead. In chunky EM, the speedup is obtained by using a variational distribution with shared (variational) membership probabilities for blocks of data (in an exhaustive partition for the entire data into non-overlapping blocks of data). The chunky EM starts from a coarse partition of the data and gradually refines the partition until convergence.

However, chunky EM does not scale well in the number of components, since all components share the *same* partition. The individual components are different – in order to obtain membership probabilities of appropriate quality, one component may need fine-grained blocks in one area of the data space, while another component is perfectly fine with coarse blocks in that area. Chunky EM expands the *shared* partition to match the needed granularity for the most demanding mixture component in any area of the data space, which might unnecessarily increase the computational

cost. Here, we derive a principled variation, called component-specific EM (CS-EM) that allows *component-specific* partitions. We demonstrate a significant performance improvement over standard and chunky EM for experiments on synthetic and mentioned customer-business data.

## 2 Background: Variational and Chunky EM

**Variational EM.** Given a set of i.i.d. data $\mathbf{x} \triangleq \{x_1, \cdots, x_N\}$, we are interested in estimating the parameters $\theta = \{\eta_{1:K}, \pi_{1:K}\}$ in the $K$-component mixture model with log-likelihood function

$$\mathcal{L}(\theta) = \sum_n \log \sum_k p(x_n|\eta_k)\pi_k. \tag{1}$$

For this task, we consider a variational generalization [11] of standard EM [4], which maximizes a lower bound of $\mathcal{L}(\theta)$ through the introduction of a variational distribution $q$. We assume that the variational distribution factorizes in accordance with data points, i.e, $q = \prod_n q_n$, where each $q_n$ is an arbitrary discrete distribution over mixture components $k = 1, \ldots, K$. We can lower bound $\mathcal{L}(\theta)$ by multiplying each $p(x_n|\eta_k)\pi_k$ in (1) with $\frac{q_n(k)}{q_n(k)}$ and apply Jensen's inequality to get

$$\mathcal{L}(\theta) \geq \sum_n \sum_k q_n(k)[\log p(x_n|\eta_k)\pi_k - \log q_n(k)] \tag{2}$$

$$= \mathcal{L}(\theta) - \sum_n \mathrm{KL}\left(q_n||p(\cdot|x_n, \theta)\right) \triangleq \mathcal{F}(\theta, q), \tag{3}$$

where $p(\cdot|x_n, \theta)$ defines the posterior distribution of membership probabilities and $\mathrm{KL}(q||p)$ is the Kullback-Leibler (KL) divergence between $q$ and $p$. The variational EM algorithm alternates the following two steps, i.e. coordinate ascent on $\mathcal{F}(\theta, q)$, until convergence.

E-step: $q^{t+1} = \arg\max_q \mathcal{F}(\theta^t, q)$,    M-step: $\theta^{t+1} = \arg\max_\theta \mathcal{F}(\theta, q^{t+1})$.

If $q$ is not restricted in any form, the E-step produces $q^{t+1} = \prod_n p(\cdot|x_n, \theta^t)$, because the KL-divergence is the only term in (3) depending on $q$. The variational EM is in this case equivalent to the standard EM, and hence produces the maximum likelihood (ML) estimate. In the following, we consider certain ways of restricting $q$ to attain speedup over standard EM, implying that the minimum KL-divergence between $q_n$ and $p(\cdot|x_n, \theta)$ is not necessarily zero. Still the variational EM defines a convergent algorithm, which instead optimizes a lower bound of the log-likelihood.

**Chunky EM.** The chunky EM algorithm [17, 16] falls into the framework of variational EM algorithms. In chunky EM, the variational distribution $q = \prod_n q_n$ is restricted according to a partition into exhaustive and mutually exclusive blocks of the data. For a given partition, if data points $x_i$ and $x_j$ are in the same block, then $q_i = q_j$. The intuition is that data points in the same block are somewhat similar and can be treated in the same way, which leads to computational savings in the E-step. If $M$ is the number of blocks in a given partition, the E-step for chunky EM has cost $O(KM)$ whereas in standard EM the cost is $O(KN)$. The speedup can be tremendous for $M \ll N$.

The speedup is gained by a trade-off between the tightness of the lower bound for the log-likelihood and the restrictiveness of constraints. Chunky EM starts from a coarse partition and iteratively refines it. This refinement process always produces a tighter bound, since restrictions on the variational distribution are gradually relaxed. The chunky EM algorithm stops when refining any block in a partition will not significantly increase the lower bound.

## 3 Component-specific EM

In chunky EM, all mixture components share the same data partition. However, for a particular block of data, the *variation* in membership probabilities differs across components, resulting in varying differences from the equality constrained variational probabilities. Roughly, the variation in membership probabilities is greatest for components closer to a block of data, and, in particular, for components far away the membership probabilities are all so small that the variation is insignificant. This intuition suggests that we might gain a computational speedup, if we create *component-specific* data partitions, where a component pays more attention to nearby data (fine-grained blocks) than data far away (coarser blocks). Let $M_k$ be the number of data blocks in the partition for component $k$. The complexity for the E-step is then $O(\sum_k M_k)$, compared to $O(KM)$ in chunky EM. Our conjecture is that we can lower bound the log-likelihood equally well with $\sum_k M_k$ significantly smaller than $KM$, resulting in a much faster E-step. Since our model maintains different partitions for different mixture components, we call it the *component-specific EM* algorithm (CS-EM).

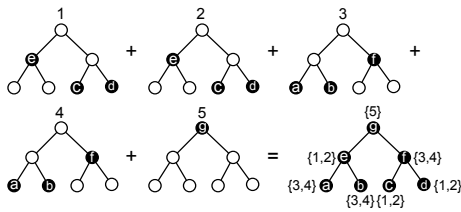

Figure 1: Trees 1-5 represent 5 mixture components with individual tree-consistent partitions ($\mathcal{B}_1$-$\mathcal{B}_5$) indicated by the black nodes. The bottom-right figure is the corresponding MPT, where $\{\cdot\}$ indicates the component marks and $a, b, c, d, e, f, g$ enumerate all the marked nodes. This MPT encodes all the component-specific information for the 5 mixtures.

**Main Algorithm.** Figure 2 (on p. 6) shows the main flow of CS-EM. Starting from a coarse partition for each component (see Section 4.1 for examples), CS-EM runs variational EM to convergence and then selectively refine the component-specific partitions. This process continues until further refinements will not significantly improve the lower bound. Sections 3.1-3.5 provide a detailed description of basic concepts in support of this brief outline for the main structure of the algorithm.

## 3.1 Marked Partition Trees

It is convenient to organize the data into a pre-computed *partition tree*, where a node in the tree represents the union of the data represented by its children. Individual data points are not actually stored in each node, but rather, the sufficient statistics necessary for our estimation operations are pre-computed and stored here. (We discuss these statistics in Section 3.3.) Any hierarchical decomposition of data that ensures some degree of similarity between data in a block is suitable for constructing a partition tree. We exemplify our work by using KD-trees [9]. Creating a KD-tree and storing the sufficient statistics in its nodes has cost $O(N \log N)$, where $N$ is the number of data point. We will in the following consider *tree-consistent partitions*, where each data block in a partition corresponds to exactly one node for a cut (possibly across different levels) in the tree–see Figure 1.

Let us now define a *marked partition tree* (MPT), a simple encoding of all component-specific partitions, as follows. Let $\mathcal{B}_k$ be the data partition (a set of blocks) in the tree-consistent partition for mixture component $k$. In Figure 1, for example, $\mathcal{B}_1$ is the partition into data blocks associated with nodes $\{e, c, d\}$. In the shared data partition tree used to generate the component-specific partitions, we mark the corresponding nodes for the data blocks in each $\mathcal{B}_k$ by the component identifier $k$. Each node $v$ in the tree will in this way contain a (possibly empty) set of component marks, denoted by $\mathcal{K}_v$. The MPT is now the subtree obtained by pruning all unmarked nodes without marked descendants from the tree. Figure 1 shows an example of a MPT. This example is special in the sense that all nodes in the MPT are marked. In general, a MPT may have unmarked nodes at any location above the leaves. For example, in chunky EM, the component-specific partitions are the same for each mixture component. In this case, only the leaves in the MPT are marked, with each leaf marked by all mixture components. The following important property for a MPT holds since all component-specific partitions are constructed with respect to the *same* data partition tree.

**Property 1.** *Let $\mathcal{T}$ denote a MPT. The marked nodes on a path from leaf to root in $\mathcal{T}$ mark exactly one data block from each of the $K$ component-specific data partitions.*

In the following, it becomes important to identify the data block in a component-specific partition, which embeds the block defined by a leaf. Let $\mathcal{L}$ denote the set of leaves in $\mathcal{T}$, and let $\mathcal{B}_{\mathcal{L}}$ denote a partition with data blocks $B_l \in \mathcal{B}_{\mathcal{L}}$ according to these leaves. We let $B_{k(l)}$ denote the specific $B_k \in \mathcal{B}_k$ with the property that $B_l \subseteq B_k$. Property 1 ensures that $B_{k(l)}$ exists for all $l, k$.

*Example:* In Figure 1, the path $a \to e \to g$ in turn marks the components $\mathcal{K}_a = \{3, 4\}$, $\mathcal{K}_e = \{1, 2\}$, and $\mathcal{K}_g = \{5\}$ and we see that each component is marked exactly once on this path, as stated in Property 1. Accordingly, for the leaf $a$, $(B_{3(a)} = B_{4(a)}) \subseteq (B_{1(a)} = B_{2(a)}) \subseteq B_{5(a)}$. □

## 3.2 The Variational Distribution

Our variational distribution $q$ assigns the same variational membership probability to mixture component $k$ for all data points in a component-specific block $B_k \in \mathcal{B}_k$. That is,

$$q_n(k) = q_{B_k} \text{ for all } x_n \in B_k, \tag{4}$$

which we denote as the *component-specific block constraint*. Unlike chunky EM, we do not assume that the data partition $\mathcal{B}_k$ is the same across different mixture components. The extra flexibility complicates the estimation of $q$ in the E-step. This is the central challenge of our algorithm.

To further drive intuition behind the E-step complication, let us make the sum-to-one constraint for the variational distributions $q_n(\cdot)$ explicit. That is, $\sum_k q_n(k) = 1$ for all data points $n$, which according to the above block constraint and using Property 1 can be reformulated as the $|\mathcal{L}|$ constraints

$$\sum_k q_{B_{k(l)}} = 1 \text{ for all } l \in \mathcal{L}. \tag{5}$$

Notice that since $q_{B_k}$ can be associated with an internal node in $\mathcal{T}$ it may be the case that $q_{B_{k(l)}}$ represent the same $q_{B_k}$ across different constraints in (5). In fact,

$$q_{B_{k(l)}} = q_{B_k} \text{ for all } l \in \{l \in \mathcal{L} | B_l \subseteq B_k\}, \tag{6}$$

implying that the constraints in (5) are intertwined according to the nested structure given by $\mathcal{T}$. The closer a data block $B_k$ is to the root of $\mathcal{T}$ the more constraints simultaneously involve the same $q_{B_k}$.

*Example:* Consider the MPT in Figure 1. Here, $q_{B_{5(a)}} = q_{B_{5(b)}} = q_{B_{5(c)}} = q_{B_{5(d)}}$, and hence the density for component 5 is the same across all four sum-to-one constraints. Similarly, $q_{B_{1(a)}} = q_{B_{1(b)}}$, so the density is the same for component 1 in the two constraints associated with leaves $a$ and $b$. $\square$

### 3.3 Efficient Variational E-step

Accounting for the component-specific block constraint in (4), the lower bound, $\mathcal{F}(\theta, q)$, in Eq. (2) can be expressed as a sum of local parts, $\mathcal{F}(\theta, q_{B_k})$, as follows

$$\mathcal{F}(\theta, q) = \sum_k \sum_{B_k \in \mathcal{B}_k} |B_k| q_{B_k} (g_{B_k} + \log \pi_k - \log q_{B_k}) = \sum_k \sum_{B_k \in \mathcal{B}_k} \mathcal{F}(\theta, q_{B_k}), \tag{7}$$

where we have defined the block-specific geometric mean

$$g_{B_k} = \langle \log p(x|\eta_k) \rangle_{B_k} = \sum_{x \in B_k} \log p(x|\eta_k)/|B_k|. \tag{8}$$

We integrate the sum-to-one constraints in (5) into the lower bound in (7) by using the standard principle of Lagrange duality (see, e.g., [1]). Accordingly, we construct the Lagrangian

$$\mathcal{F}(\theta, q, \lambda) = \sum_k \sum_{B_k} \mathcal{F}(\theta, q_{B_k}) + \sum_l \lambda_l (\sum_k q_{B_{k(l)}} - 1),$$

where $\lambda \triangleq \{\lambda_1, \ldots, \lambda_L\}$ are the Lagrange multipliers for the constraints in Eq. (5). Recall the relationship between $q_{B_k}$ and $q_{B_{k(l)}}$ in (6). By setting $\partial \mathcal{F}(\theta, q, \lambda)/\partial q_{B_k} = 0$, we obtain

$$q_{B_k}(\lambda) = \exp\left((1/|B_k|) \sum_{l:B_l \subseteq B_k} \lambda_l - 1\right) \pi_k \exp(g_{B_k}). \tag{9}$$

Solving the dual optimization problem $\lambda^* = \arg\min_\lambda \mathcal{F}(\theta, q(\lambda), \lambda)$ now leads to the primal solution given by $q_{B_k}^* = q_{B_k}(\lambda^*)$.[1]

For chunky EM, the E-step is straightforward, because $B_{k(l)} = B_l$ and therefore $\sum_{l:B_l \subseteq B_{k(l)}} \lambda_l = \lambda_l$ for all $k = 1, \ldots, K$. Substituting (9) into the sum-to-one constraints in (5) reveals that each $\lambda_l$ can be solved independently, leading to the following closed-form solution for $q_{B_{k(l)}}$

$$\lambda_l^* = |B_l| \left(1 + \log \sum_k \pi_k \exp(g_{B_{k(l)}})\right), \quad q_{B_{k(l)}}^* = \pi_k \exp(g_{B_{k(l)}})/Z, \tag{10}$$

where $Z = \sum_k \pi_k \exp(g_{B_{k(l)}})$ is a normalizing constant.

CS-EM does not enjoy a similar simple optimization, because of the intertwined constraints, as described in Section 3.2. Fortunately, we can still obtain a closed-form solution. Essentially, we use the nesting structure of the constraints to reduce Lagrange multipliers from the solution one at a time until only one is left, in which case the optimization is easily solved. We describe the basic approach here and defer the technical details (and pseudo-code) to the supplement.

Consider a leaf node $l \in \mathcal{L}$ and recall that $\mathcal{K}_l$ denotes the components with $B_{k(l)} = B_l$ in their partitions. The sum-to-one constraint in (5) that is associated with leaf $l$ can therefore be written as

$$\sum_{k \in \mathcal{K}_l} q_{B_{k(l)}} + \sum_{k \notin \mathcal{K}_l} q_{B_{k(l)}} = 1.$$

Furthermore, for all $k \in \mathcal{K}_l$ the $q_{B_{k(l)}}$, as defined in (9), is a function of the same $\lambda_l$. Accordingly,

$$q_l \triangleq \sum_{k \in \mathcal{K}_l} q_{B_{k(l)}} = \exp(\lambda_l/|B_l| - 1) \sum_{k \in \mathcal{K}_l} \pi_k \exp(g_{B_{k(l)}}). \tag{11}$$

Now, consider $l$'s leaf-node sibling, $l'$. For example, in Figure 1, node $l = a$ and $l' = b$. The two leaves share the same path from their parent to the root in $\mathcal{T}$. Hence, using Property 1, it must be the case that $B_{k(l)} = B_{k(l')}$ for $k \notin \mathcal{K}_l$. The two sum-to-one constraints–one for each leaf–therefore imply that $q_l = q_{l'}$. Using (11), it now follows that

$$\lambda_{l'} = |B_{l'}|(\lambda_l/|B_l|) + \log \sum_{k \in \mathcal{K}_l} \pi_k \exp(g_{B_{k(l)}}) - \log \sum_{k \in \mathcal{K}_{l'}} \pi_{k'} \exp(g_{B_{k(l')}})) \triangleq f(\lambda_l).$$

Thus, we can replace $\lambda_{l'}$ with $f(\lambda_l)$ in all $q_{B_k}$ expressions. Further analysis (detailed in the supplement) shows how we more efficiently account for this parameter reduction and continue the process, now considering the parent node a new "leaf" node once all children have been processed. When reaching the root, every $q_{B_k}$ expression on the path from $l$ only involves the single $\lambda_l$, and the optimal $\lambda_l^*$ can therefore be found analytically by solving the corresponding sum-to-one constraint in (5). Following, all optimal $q_{B_k}^*$ are found by inserting $\lambda_l^*$ into the reduced $q_{B_k}$ expressions.

Finally, it is important to notice that $g_{B_k}$ is the only data-dependent part in the above E-step solution. It is therefore key to the computational efficiency of the CS-EM algorithm that $g_{B_k}$ can be calculated from pre-computed statistics, which is in fact the case for the large class of exponential family distributions. These are the statistics that are stored in the nodes of the MPT.

*Example:* Let $p(x|\eta_k)$ be an exponential family distribution

$$p(x|\eta_k) = h(x)\exp(\eta_k^T T(x) - A(\eta_k)), \tag{12}$$

where $\eta_k$ is the natural parameter, $h(x)$ is the reference function, $T(x)$ is the sufficient statistic, and $A(\eta_k)$ is the normalizing constant. Then

$$g_{B_k} = \langle \log h(x)\rangle_{B_k} + \eta_k^T \langle T(x)\rangle_{B_k} - A(\eta_k),$$

where $\langle \log h(x)\rangle_{B_k}$ and $\langle T(x)\rangle_{B_k}$ are the statistics that we pre-compute for (8). In particular, if $p(x|\eta_k) = \mathcal{N}_d(\mu_k, \Sigma_k)$, a Gaussian distribution, then

$$h(x)=1,\ \ T(x)=(x,\ xx^T),\ \ \eta_k=(\mu_k\Sigma_k^{-1}, -\Sigma_k^{-1}/2),\ \ A(\eta_k) = -\tfrac{1}{2}\big(d\log(2\pi)+\log|\Sigma_k|+\mu_k^T\Sigma^{-1}\mu_k\big),$$

and the statistics $\langle \log h(x)\rangle_{B_k} = 0$ and $\langle T(x)\rangle_{B_k} = (\langle x\rangle_{B_k}, \langle xx^T\rangle_{B_k})$ can be pre-computed. □

### 3.4 Efficient Variational M-step

In the variational M-step the model parameters $\theta = \{\eta_{1:K}, \pi_{1:K}\}$ are updated by maximizing Eq. (7) w.r.t. $\theta$ under the constraint $\sum_k \pi_k = 1$. Hereby, the update is

$$\pi_k \propto \sum_{B_k \in \mathcal{B}_k} |B_k| q_{B_k},\ \ \eta_k = \arg\max_{\eta_k} \sum_{B_k \in \mathcal{B}_k} |B_k| q_{B_k} g_{B_k}. \tag{13}$$

Thus, the M-step can be efficiently computed using the pre-computed sufficient statistics as well.

*Example:* If $p(x|\eta_k)$ has the exponential family form in Eq. (12), $\eta_k$ is obtained by solving

$$\eta_k = \arg\max_{\eta_k} (\sum_{B_k \in \mathcal{B}_k} q_{B_k} \sum_{x \in B_k} T(x))\eta_k - (\sum_{B_k \in \mathcal{B}_k} |B_k| q_{B_k}) A(\eta_k).$$

In particular, if $p(x|\eta_k) = \mathcal{N}_d(\mu_k, \Sigma_k)$, then

$$\mu_k = (\sum_{B_k \in \mathcal{B}_k} |B_k| q_{B_k} \langle x\rangle_{B_k})/(N\pi_k),\ \ \Sigma_k = (\sum_{B_k \in \mathcal{B}_k} |B_k| q_{B_k} \langle xx^T\rangle_{B_k} - \mu_k\mu_k^T)/(N\pi_k). \ □$$

### 3.5 Efficient Variational R-step

Given the current component-specific data partitions, as marked in the MPT $\mathcal{T}$, a refining step (R-step) selectively refines these partitions. Any refinement enlarges the family of variational distributions, and therefore always tightens the optimal lower bound for the log-likelihood. We define a *refinement unit* as the refinement of *one* data block in the current partition for *one* component in the model. The efficiency of CS-EM is affected by the number of refinement units performed at each R-step. With too few units we spend too much time on refining, and with too many units some of the refinements may be far from optimal and therefore unnecessarily slow down the algorithm. We have empirically found $K$ refinement units at each R-step to be a good choice. This introduces $K$ new free variational parameters, which is similar to a refinement step in chunky EM. However, chunky EM refines the same data block across all components, which is not the case in CS-EM.

| Figure 2: The CS-EM algorithm. | Figure 3: Variational R-step algorithm. |
|---|---|
| 1: *Initialization*: build KD-tree, set initial MPT, set initial $\theta$, run E-step to set $q$, set $t, s = 0$, compute $\mathcal{F}_t, \mathcal{F}_s$ using (7). | 1: Initialize priority queue $Q$ favoring high $\Delta\mathcal{F}_{v,k}$ values. |
| 2: **repeat** | 2: **for** each marked node $v$ in $\mathcal{T}$ **do** |
| 3:   **repeat** | 3:   Compute $q$ via E-step with constraints as in (14). |
| 4:     Run variational E-step and M-step. | 4:   **for all** $k \in \mathcal{K}_v$ **do** |
| 5:     Set $t \leftarrow t + 1$ and compute $\mathcal{F}_t$ using (7). | 5:     Insert candidate $(v, k)$ into $Q$ according to $\Delta\mathcal{F}_{v,k}$. |
| 6:   **until** $(\mathcal{F}_t - \mathcal{F}_{t-1})/(\mathcal{F}_t - \mathcal{F}_0) < 10^{-4}$. | 6:   **end for** |
| 7:   Run variational R-step. | 7: **end for** |
| 8:   Set $s \leftarrow s + 1$ and $\mathcal{F}_s = \mathcal{F}_t$. | 8: Select $K$ top-ranked $(v, k)$ in $Q$ for refinement. |
| 9: **until** $(\mathcal{F}_s - \mathcal{F}_{s-1})/(\mathcal{F}_s - \mathcal{F}_0) < 10^{-4}$. | |

Ideally, an R-step should select the refinement units leading to optimal improvement for $\mathcal{F}$. Good candidates can be found by performing a single E-step for each candidate and then select the units that improve $\mathcal{F}$ the most. This demands the evaluation of an E-step for each of the $\sum_k M_k$ possible refinement units. Exact evaluation for this many full E-steps is prohibitively expensive, and we therefore instead approximate these refinement-guiding E-steps by a local computation scheme based on the intuition that refining a block for a specific component mostly affects components with similar local partition structures. The algorithm is described in Figure 3 with details as follows.

Consider moving all component-marks for $v \in \mathcal{T}$ to its children $ch(v)$, where each child $u \in ch(v)$ receives a copy. Let $\bar{\mathcal{T}}$ denote the altered MPT, and $\bar{\mathcal{K}}_v, \bar{\mathcal{K}}_u$ denote the set of marks at $v, u \in \bar{\mathcal{T}}$. Hence, $\bar{\mathcal{K}}_v = \emptyset$ and $\bar{\mathcal{K}}_u = \mathcal{K}_u \cup \mathcal{K}_v$. To approximate the new variational distribution $\bar{q}$, we fix the value for each $\bar{q}_{B_{k(l)}}$, with $k \notin \bar{\mathcal{K}}_u$ and $l \in \mathcal{L}$, to the value obtained for the distribution $q$ before the refinement. In this case, the sum-to-one constraints for $\bar{q}$ simplifies as

$$\sum_{k \in \bar{\mathcal{K}}_u} \bar{q}_{B_{k(l)}} + R_l = 1 \text{ for all } l \in \mathcal{L}, \tag{14}$$

with $R_l = 1 - \sum_{k \in \bar{\mathcal{K}}_u} q_{B_{k(l)}}$ being the fixed values. Notice that $\sum_{k \in \bar{\mathcal{K}}_u} q_{B_{k(l)}} = 0$ for any leaf $l$ not under $u$, and that $q_{B_{k(l)}} = q_{B_{k(u)}}$ and $\bar{q}_{B_{k(l)}} = \bar{q}_{B_{k(u)}}$ for $k \in \bar{\mathcal{K}}_u$ and any leaf $l$ under $u$. The constraints in (14) therefore reduces to the following $|ch(v)|$ independent constraints

$$\sum_{k \in \bar{\mathcal{K}}_u} \bar{q}_{B_{k(u)}} + R_u = 1 \text{ for all } u \in ch(v).$$

Each $\bar{q}_{B_{k(u)}}, k \in \bar{\mathcal{K}}_u$ now has a local closed form solution similar to (10)–with $Z = \sum_{k \in \bar{\mathcal{K}}_u} \bar{q}_{B_{k(u)}} + R_u$.

The improvement to $\mathcal{F}$ that is achieved by the refinement-guiding E-step for the refinement unit refining data block $v$ for component $k$ is denoted $\Delta\mathcal{F}_{v,k}$, and can be computed as

$$\Delta\mathcal{F}_{v,k} = \sum_{u \in ch(v)} \mathcal{F}(\theta, \bar{q}_{B_{k(u)}}) - \mathcal{F}(\theta, q_{B_{k(v)}}).$$

This improvement is computed for all possible refinement units and the $K$ highest scoring units are then selected in the R-step. Notice that this selective refinement step will most likely not refine the same data block for all components and therefore creates component-specific partitions.

*Example:* In Figure 1, node $e$ and its children $\{a, b\}$ are marked $K_e = \{1, 2\}$ and $\mathcal{K}_a = \mathcal{K}_b = \{3, 4\}$. For the two candidate refinement units associated with $e$, we have $\bar{\mathcal{K}}_e = \emptyset$ and $\bar{\mathcal{K}}_a = \bar{\mathcal{K}}_b = \{1, 2, 3, 4\}$. With $q_{5(u)}$ held fixed, we will for each child $u \in \{a, b\}$ optimize $\bar{q}_{B_{k(u)}}, k = 1, 2, 3, 4$, and following $(e, 1)$ and $(e, 2)$ are inserted into the priority queue of candidates according to their $\Delta\mathcal{F}_{v,k}$ values. □

## 4 Experiments

In this section we provide a systematic evaluation of CS-EM, chunky EM, and standard EM on synthetic data, as well as a comparison between CS-EM and chunky EM on the business-customer data, mentioned in Section 1. (Standard EM is too slow to be included in the latter experiment.)

### 4.1 Experimental setup

For the synthetic experiments, we generated random training and test data sets from Gaussian mixture models (GMMs) by varying one (in a single case two) of the following default settings: #data points $N = 100,000$, #mixture components $K = 40$, #dimensions $d = 2$, and c-separation[2] $c = 2$.

The (proprietary) business-customer data was obtained through collaboration with PitneyBowes Inc. and Yellowpages.com LLC. For the experiments on this data, $N = 6.5$ million and $d = 2$, corresponding to the latitude and longitude for potential customers in Washington state. The basic assumption is that potential customers act as rational consumers and frequent the somewhat closest business locations to purchase a good or service. The locations for competing stores of a particular type, in this way, correspond to fixed centers for components in a mixture model. (A less naive model with the penetration level for a good or service and the relative attractiveness for stores, is the object of related research, but is not important for the computational feasibility studied here.)

The synthetic experiments are initialized as follows. After constructing KD-tree, the first tree-level containing at least $K$ nodes ($\lceil \log_2 K \rceil$) is used as the initial data partition for both chunky EM and all components in CS-EM. For all algorithms (including standard EM), we randomly chose $K$ data blocks from the initial partition and initialized parameters for the individual mixture components accordingly. Mixture weights are initialized with a uniform distribution. The experiments on the business-customer data are initialized in the same way, except that the component centers are fixed and the initial data blocks that cover these centers are used for initializing the remaining parameters.

For CS-EM we also considered an alternative initialization of data partitions, which better matches the rationale behind component-specific partitions. It starts from the CS-EM initialization and recursively, according to the KD-tree structure, merges two data blocks in a component-specific partition, if the merge has little effect on that component.[3] We name this variant as *CS-EM*$^*$.

## 4.2   Results

For the synthetic experiments, we compared the run-times for the competing algorithms to reach a parameter estimate of same quality (and therefore similar clustering performance not counting different local maxima), defined as follows. We recorded the log-likelihood for the test data at each iteration of the EM algorithm, and before each S-step in chunky EM and the CS-EM. We ran all algorithms to convergence at level $10^{-4}$, and the test log-likelihood for the algorithm with lowest value was chosen as baseline.[4] We now recorded the run-time for each algorithm to reach this baseline, and computed the *EM-speedup* factors for chunky EM, CS-EM, and CS-EM$^*$, each defined as the standard EM run-time divided by the run-time for the alternative algorithm. We repeated all experiments with five different parameter initializations and report the averaged results.

Figure 4 shows the EM-speedups for the synthetic data. First of all, we see that both CS-EM and CS-EM$^*$ are significantly faster than chunky EM in all experiments. In general, the $\sum_k M_k$ variational parameters needed for the CS-EM algorithms is far fewer than the $KM$ parameters needed for chunky EM in order to reach an estimate of same quality. For example, for the default experimental setting, the ratio $KM/\sum_k M_k$ is 2.0 and 2.1 for, respectively, CS-EM and CS-EM$^*$. We also see that there is no significant difference in speedup between CS-EM and CS-EM$^*$. This observation can be explained by the fact that the resulting component-specific data partitions greatly refine the initial partitions, and any computational speedup due to the smarter initial partition in CS-EM$^*$ is therefore overwhelmed. Hence, a simple initial partition, as in CS-EM, is sufficient.

Finally, similar to results already reported for chunky EM in [17, 16], we see for all of chunky EM, CS-EM, and CS-EM$^*$ that the number of data points and the amount of $c$-separation have a positive effect on EM-speedup, while the number of dimensions and the number of components have a negative effect. However, the last plot in Figure 4 reveals an important difference between chunky EM and CS-EM: with a fixed ratio between number of data points and number of clusters, the EM-speedup declines a lot for chunky EM, as the number of clusters and data points increases. This observation is important for the business-customer data, where increasing the area of investigation (from city to county to state to country) has this characteristic for the data.

In the second experiment on the business-customer data, standard EM is computationally too demanding. For example, for the "Nail salon" example in Figure 5, a single EM iteration takes about 5 hours. In contrast, CS-EM runs to convergence in 20 minutes. To compare run-times for chunky

Figure 4: EM-speedup factors on synthetic data.

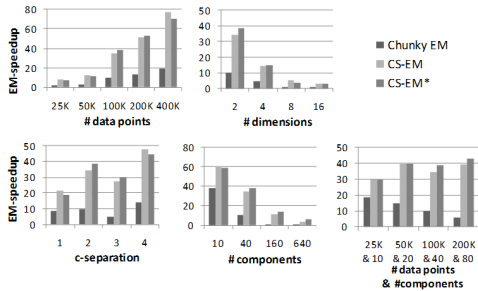

Figure 5: A comparison of run-time and final number of variational parameters for Chunky EM vs. CS-EM for exemplary business types with different number of stores (mixture components).

| Business type | #stores | time ratio | parameter ratio |
|---|---|---|---|
| Bowling | 129 | 5.0 | 2.41 |
| Dry cleaning | 815 | 21.2 | 2.81 |
| Nail salon | 1290 | 35.8 | 3.51 |
| Pizza | 1327 | 33.0 | 3.18 |
| Tax filing | 1459 | 34.8 | 3.41 |
| Conv. store | 1739 | 29.4 | 3.42 |

EM and CS-EM, we therefore slightly modified the way we ensure that the two algorithm reach a parameter estimate of same quality. We use the lowest of the $\mathcal{F}$ values (on training data) obtained for the two algorithms at convergence as the baseline, and record the time for each algorithm to reach this baseline. Figure 5 shows the speedup (time ratio) and the reduction in number of variational parameters (parameter ratio) for CS-EM compared to chunky EM, as evaluated on exemplary types of businesses. Again, CS-EM is significantly faster than chunky EM and the speedup is achieved by a better targeting of variational distribution through the component-specific partitions.

## 5 Related and Future Work

**Related work.** CS-EM combines the best from two major directions in the literature regarding speedup of EM for mixture modeling. The first direction is based on powerful heuristic ideas, but without provable convergence due to the lack of a well-defined objective function. The work in [10] is a prominent example, where KD-tree partitions were first used for speeding up EM. As also pointed out in [17, 16], the method will likely–but not provably–converge for fine-grained partitions. In contrast, CS-EM is provable convergent–even for arbitrary rough partitions, if extreme speedup is needed. The granularity of partitions in [10] is controlled by a user-specified threshold on the minimum and maximum membership probabilities that are reachable within the boundaries of a node in the KD-tree. In contrast, we have almost no tuning parameters. We instead let the data speak by itself by having the final convergence determine the granularity of partitions. Finally, [10] "prunes" a component (sets the membership probability to zero) for data far away from the component. It relates to our component-specific partitions, but ours is more principled with convergence guarantees.

The second direction of speedup approaches are based on the variational EM framework [11]. In [11], a "sparse" EM was presented, which at some iterations, only updates part of the parameters and hence relates it to the pruning idea in [10]. [14] presents an "incremental" and a "lazy" EM, which gain speedup by performing E-steps on varying subsets of the data rather than the entire data. All three methods guarantee convergence. However, they need to periodically perform an E-step over the entire data, and, in contrast to CS-EM, their E-step is therefore not truly sub-linear in sample size, making them potentially unsuitable for large-scale applications. The chunky EM in [17, 16] is the approach most similar to our CS-EM. Both are based on the variational EM framework and therefore guarantees convergence, but CS-EM is faster and scales better in the number of clusters.

In addition, heuristic sub-sampling is common practice when faced with a large amount of data. One could argue that chunky EM is an intelligent sub-sampling method, where 1) instead of sampled data points it uses geometric averages for blocks of data in a given data partition, and 2) it automatically chooses the "sampling size" by a learning curve method, where $\mathcal{F}$ is used to measure the utility of increasing the granularity for the partition. Sub-sampling therefore has same computational complexity as chunky EM, and our results therefore suggest that we should expect CS-EM to be much faster than sub-sampling and scale better in the number of mixture components.

Finally, we exemplified our work by using KD-trees as the tree-consistent partition structure for generating the component-specific partitions in CS-EM, which limited its effectiveness in high dimensions. However, any hierarchical partition structure can be used, and the work in [8] therefore suggest that changing to an anchor tree (a special kind of metric tree [15]) will also render CS-EM effective in high dimensions, under the assumption of lower intrinsic dimensionality for the data.

**Future Work.** Future work will include parallelization of the algorithm and extensions to 1) non-probabilistic clustering methods, e.g., k-means clustering [6, 13, 5] and 2) general EM applications beyond mixture modeling.

## Footnotes

*Equal contributors. †Work done during internship at Microsoft Research.

[1] Notice that Eq. (9) implies that positivity constraints $q_n(k) \geq 0$ are automatically satisfied during estimation.

[2]A GMM is c-separated [3], if for any $i \neq j$, $f(i, j) \triangleq ||\mu_i - \mu_j||^2 / \max(\lambda_{\max}(\Sigma_i), \lambda_{\max}(\Sigma_j)) \geq dc^2$, where $\lambda_{\max}(\Sigma)$ denotes the maximum eigenvalue of $\Sigma$. We only require that $\text{Median}\,[f(i, j)] \geq dc^2$.

[3]Let $\mu$ and $\Sigma$ be the mean and variance parameter for an initial component, and $\mu_p$, $\mu_l$, and $\mu_r$ denote the sample mean for data in the considered parent, left and right child. We merge if $|MD(\mu_l, \mu|\Sigma)/MD(\mu_p, \mu|\Sigma) - 1| < 0.05$ and $|MD(\mu_r, \mu|\Sigma)/MD(\mu_p, \mu|\Sigma) - 1| < 0.05$, where $MD(\cdot, \cdot|\Sigma)$ is the Mahalanobis distance.

[4]For the default experimental setting, for example, the baseline is reached at 96% of the log-likelihood improvement from initialization to standard EM convergence.

# References

[1] S. Boyd and L. Vandenberghe. *Convex Optimization*. Cambridge University Press, 2004.

[2] P. S. Bradley, U. M. Fayyad, and C. A. Reina. Scaling EM (expectation maximization) clustering to large databases. Technical Report MSR-TR-98-3, Microsoft Research, 1998.

[3] S. Dasgupta. Learning mixtures of Gaussians. In *Proceedings of the 40th Annual Symposium on Foundations of Computer Science*, pages 634–644, 1999.

[4] A. P. Dempster, N. M. Laird, and D. B. Rubin. Maximum likelihood from incomplete data via the EM algorithm. *Journal of the Royal Statistical Society, Series B*, 39(1):1–38, 1977.

[5] G. Hamerly. Making k-means even faster. In *SIAM International Conference on Data Mining (SDM)*, 2010.

[6] T. Kanungo, D. M. Mount, N. S. Netanyahu, C. D. Piatko, R. Silverman, and A. Y. Wu. An efficient k-means clustering algorithm: Analysis and implementation. *IEEE Transactions on Pattern Analysis and Machine Intelligence*, 24(7):881–892, 2002.

[7] G. J. McLachlan and D. Peel. *Finite Mixture Models*. Wiley Interscience, New York, USA, 2000.

[8] A. Moore. The anchors hierarchy: Using the triangle inequality to survive high-dimensional data. In *Proceedings of the Fourteenth Conference on Uncertainty in Artificial Intelligence*, pages 397–405. AAAI Press, 2000.

[9] A. W. Moore. A tutorial on kd-trees. Technical Report 209, University of Cambridge, 1991.

[10] A. W. Moore. Very fast EM-based mixture model clustering using multiresolution kd-trees. In *Advances in Neural Information Processing Systems*, pages 543–549. Morgan Kaufman, 1999.

[11] R. Neal and G. E. Hinton. A view of the EM algorithm that justifies incremental, sparse, and other variants. In *Learning in Graphical Models*, pages 355–368, 1998.

[12] L. E. Ortiz and L. P. Kaelbling. Accelerating EM: An empirical study. In *Proceedings of the Fifteenth Conference on Uncertainty in Artificial Intelligence*, pages 512–521, 1999.

[13] D. Pelleg and A. Moore. Accelerating exact k-means algorithms with geometric reasoning. In S. Chaudhuri and D. Madigan, editors, *Proceedings of the Fifth International Conference on Knowledge Discovery in Databases*, pages 277–281. AAAI Press, 1999.

[14] B. Thiesson, C. Meek, and D. Heckerman. Accelerating EM for large databases. *Machine Learning*, 45(3):279–299, 2001.

[15] J. K. Uhlmann. Satisfying general proximity/similarity queries with metric trees. *Information Processing Letters*, 40(4):175–179, 1991.

[16] J. J. Verbeek, J. R. Nunnink, and N. Vlassis. Accelerated EM-based clustering of large data sets. *Data Mining and Knowledge Discovery*, 13(3):291–307, 2006.

[17] J. J. Verbeek, N. Vlassis, and J. R. J. Nunnink. A variational EM algorithm for large-scale mixture modeling. In *In Proceedings of the 8th Annual Conference of the Advanced School for Computing and Imaging (ASCI)*, 2003.

